# Minkowski-r Back-Propagation: Learning in Connectionist Models with Non-Euclidian Error Signals

Stephen José Hanson and David J. Burr

Bell Communications Research
Morristown, New Jersey 07960

## Abstract

Many connectionist learning models are implemented using a gradient descent in a least squares error function of the output and teacher signal. The present model generalizes, in particular, back-propagation [1] by using Minkowski-r power metrics. For small $r$'s a "city-block" error metric is approximated and for large $r$'s the "maximum" or "supremum" metric is approached, while for $r=2$ the standard back-propagation model results. An implementation of Minkowski-r back-propagation is described, and several experiments are done which show that different values of $r$ may be desirable for various purposes. Different r values may be appropriate for the reduction of the effects of outliers (noise), modeling the input space with more compact clusters, or modeling the statistics of a particular domain more naturally or in a way that may be more perceptually or psychologically meaningful (e.g. speech or vision).

## 1. Introduction

The recent resurgence of connectionist models can be traced to their ability to do complex modeling of an input domain. It can be shown that neural-like networks containing a single hidden layer of non-linear activation units can learn to do a *piece-wise linear* partitioning of a feature space [2]. One result of such a partitioning is a complex gradient surface on which decisions about new input stimuli will be made. The generalization, categorization and clustering properties of the network are therefore determined by this mapping of input stimuli to this gradient surface in the output space. This gradient surface is a function of the conditional probability distributions of the output vectors given the input feature vectors as well as a function of the error relating the teacher signal and output.

Presently many of the models have been implemented using least squares error. In this paper we describe a new model of gradient descent back-propagation [1] using Minkowski-r power error metrics. For small r's a "city-block" error measure (r=1) is approximated and for larger r's a "maximum" or supremum error measure is approached, while the standard case of Euclidian back-propagation is a special case with r=2. First we derive the general case and then discuss some of the implications of varying the power in the general metric.

## 2. Derivation of Minkowski-r Back-propagation

The standard back-propagation is derived by minimizing least squares error as a function of connection weights within a completely connected layered network. The error for the Euclidian case is (for a single input-output pair),

$$E = \frac{1}{2} \sum_i (y_i - \hat{y}_i)^2,$$ (1)

where $y$ is the activation of a unit and $\hat{y}$ represents an independent teacher signal. The activation of a unit ($y$) is typically computed by normalizing the input from other units ($x$) over the interval (0,1) while compressing the high and low end of this range. A common function used for this normalization is the logistic,

$$y_i = \frac{1}{1 + e^{-x_i}}$$ (2)

The input to a unit ($x$) is found by summing products of the weights and corresponding activations from other units,

$$x_i = \sum_h y_h w_{hi},$$ (3)

where $y_h$ represents units in the fan in of unit $i$ and $w_{hi}$ represents the strength of the connection between unit $i$ and unit $h$.

A gradient for the Euclidian or standard back-propagation case could be found by finding the partial of the error with respect to each weight, and can be expressed in this three term differential,

$$\frac{\partial E}{\partial w_{hi}} = \frac{\partial E}{\partial y_i}\frac{\partial y_i}{\partial x_i}\frac{\partial x_i}{\partial w_{hi}} \tag{4}$$

which from the equations before turns out to be,

$$\frac{\partial E}{\partial w_{hi}} = (y_i - \hat{y}_i)y_i(1-y_i)y_h \tag{5}$$

Generalizing the error for Minkowski-r power metrics (see Figure 1 for the family of curves),

$$E = \frac{1}{r}\sum_i |(y_i - \hat{y}_i)|^r \tag{6}$$

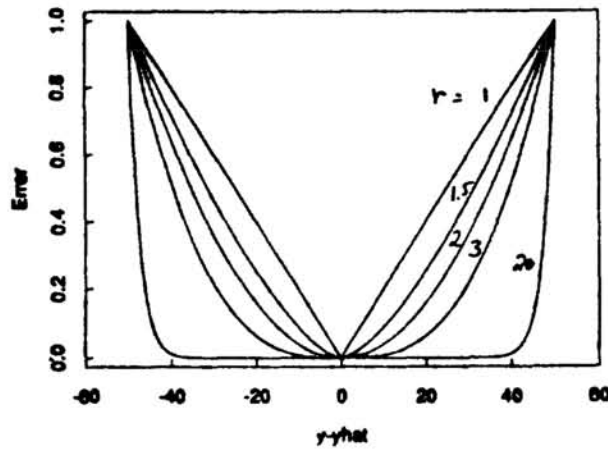

*Figure 1: Minkowski-r Family*

Using equations 2-4 above with equation 6 we can easily find an expression for the gradient in the general Minkowski-r case,

$$\frac{\partial E}{\partial w_{hi}} = (|y_i - \hat{y}_i|)^{r-1}y_i(1-y_i)y_h sgn(y_i - \hat{y}_i) \tag{7}$$

This gradient is used in the weight update rule proposed by Rumelhart, Hinton and Williams [1],

$$w_{hi}(n+1) = \alpha \frac{\partial E}{\partial w_{hi}} + w_{hi}(n) \qquad (8)$$

Since the gradient computed for the hidden layer is a function of the gradient for the output, the hidden layer weight updating proceeds in the same way as in the Euclidian case [1], simply substituting this new Minkowski-r gradient.

It is also possible to define a gradient over r such that a minimum in error would be sought. Such a gradient was suggested by White [3, see also 4] for maximum likelihood estimation of r, and can be shown to be,

$$\frac{d\log(E)}{dr} = (1-1/r)(1/r) + (1/r)^2 log(r) + (1/r)^2 \psi(1/r) + (1/r)^2 |y_i - \hat{y}_i|$$

$$-(1/r)(|y_i - \hat{y}_i|)^r log(|y_i - \hat{y}_i|) \qquad (9)$$

An approximation of this gradient (using the last term of equation 9) has been implemented and investigated for simple problems and shown to be fairly robust in recovering similar r values. However, it is important that the r update rule changes slower than the weight update rule. In the simulations we ran r was changed once for every 10 times the weight values were changed. This rate might be expected to vary with the problem and rate of convergence. Local minima may be expected in larger problems while seeking an optimal r. It may be more informative for the moment to examine different classes of problems with fixed r and consider the specific rationale for those classes of problems.

## 3. Variations in r

Various r values may be useful for various aspects of representing information in the feature domain. Changing r basically results in a reweighting of errors from output bits[1]. Small r's give less weight for large deviations and tend to reduce the influence of outlier points in the feature space during learning. In fact, it can be shown that if the distributions of feature vectors are non-gaussian, then the r=2 case

---

1. It is possible to entertain r values that are negative, which would give largest weight to small errors close to zero and smallest weight to very large errors. Values of r less than 1 generally are non-metric, i.e. they violate at least one of the metric axioms. For example, r<0 violates the triangle inequality. For some problems this may make sense and the need for a metric error weighting may be unnecessary. These issues are not explored in this paper.

will *not* be a maximum likelihood estimator of the weights [5]. The city block case, r=1, in fact, arises if the underlying conditional probability distributions are Laplace [5]. More generally, r's less than two will tend to model non-gaussian distributions where the tails of the distributions are more pronounced than in the gaussian. Better estimators can be shown to exist for general noise reduction and have been studied in the area of *robust estimation* procedures [5] of which the Minkowski-r metric is only one possible case to consider.

*r*<2. It is generally recommended that r=1.5 may be optimal for many noise reduction problems [6]. However, noise reduction may also be expected to vary with the problem and nature of the noise. One example we have looked at involves the recovery of an arbitrary 3 dimensional smooth surface as shown in Figure 2a, after the addition of random noise. This surface was generated from a gaussian curve in the 2 dimensions. Uniform random noise equal to the width (standard deviation) of the surface shape was added point-wise to the surface producing the noise plus surface shape shown in Figure 2b.

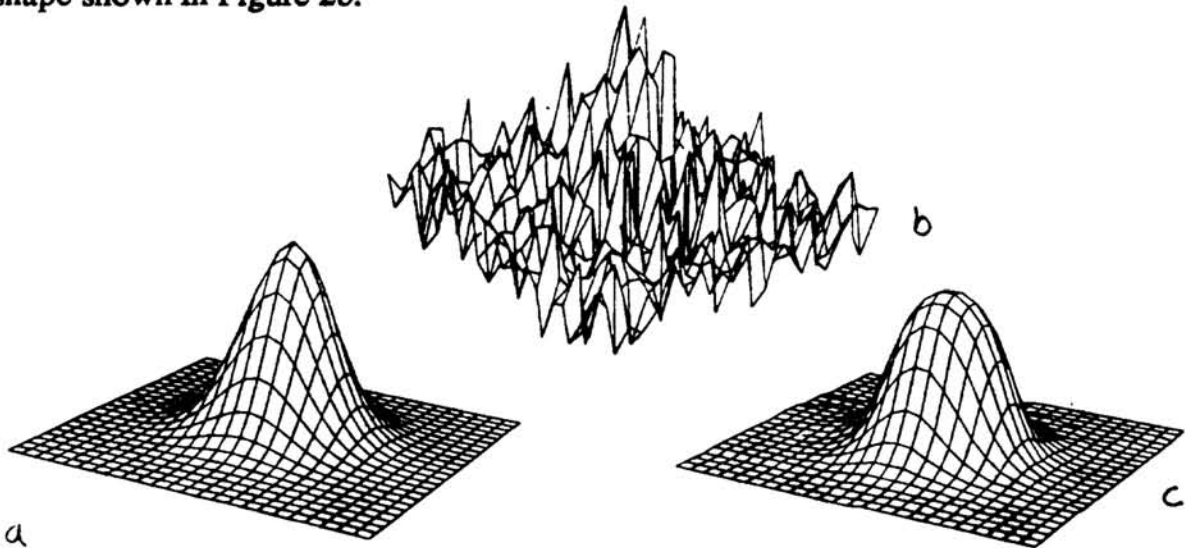

*Figure 2: Shape surface (2a), Shape plus noise surface (2b) and recovered Shape surface (2c)*

The shape in Figure 2a was used as target points for Minkowski-r back-propagation[2] and recovered with some distortion of the slope of the shape near the peak of the

---

2. All simulation runs, unless otherwise stated, used the same learning rate (.05) and smoothing value (.9) and stopping criterion defined in terms of absolute mean deviation. The number of iterations to meet the stopping criterion varied considerably as r was changed (see below).

surface (see Figure 2c). Next the noise plus shape surface was used as target points for the learning procedure with r=2. The shape shown in Figure 3a was recovered, however, with considerable distortion iaround the base and peak. The value of r was reduced to 1.5 (Figure 3b) and then finally to 1.2 (Figure 3c) before shape distortions were eliminated. Although, the major properties of the shape of the surface were recovered, the scale seems distorted (however, easily restored with renormalization into the 0,1 range).

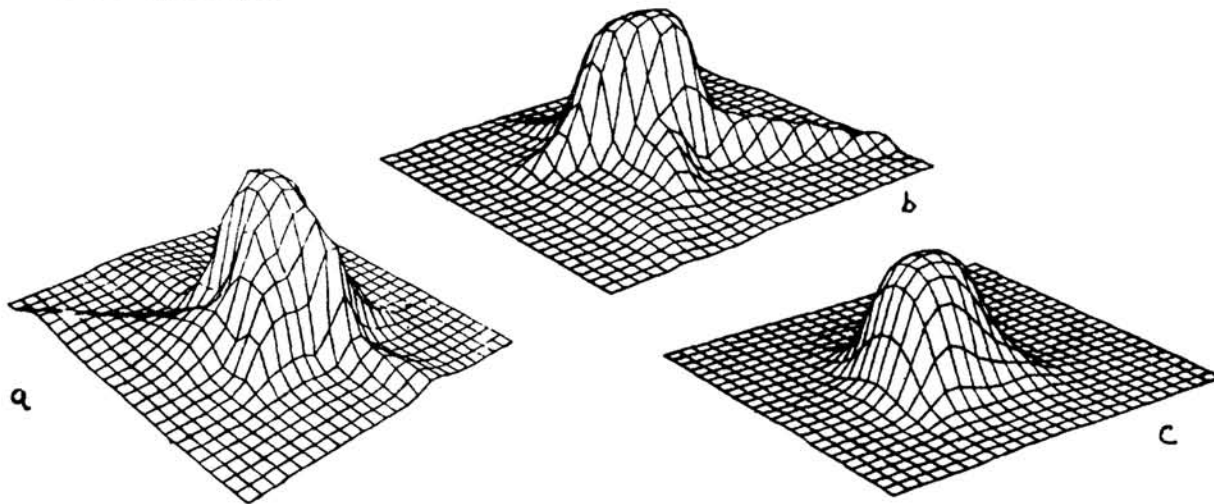

*Figure 3: Shape surface recovered with r=2 (3a), r=1.5 (3b) and r=1.2 (3c)*

$r>2$. Large r's tend to weight large deviations. When noise is not possible in the feature space (as in an arbitrary boolean problem) or where the token clusters are compact and isolated then simpler (in the sense of the number and placement of partition planes) generalization surfaces may be created with larger r values. For example, in the simple XOR problem, the main effect of increasing r is to pull the decision boundaries closer into the non-zero targets (compare high activation regions in Figure 4a and 4b).

In this particular problem clearly such compression of the target regions does not constitute simpler decision surfaces. However, if more hidden units are used than are needed for pattern class separation, then increasing r during training will tend to reduce the number of cuts in the space to the minimum needed. This seems to be primarily due to the sensitivity of the hyper-plane placement in the feature space to the geometry of the targets.

A more complex case illustrating the same idea comes from an example suggested by Minsky & Papert [7] called "the mesh". This type of pattern recognition problem is also, like XOR, a non-linearly separable problem. An optimal

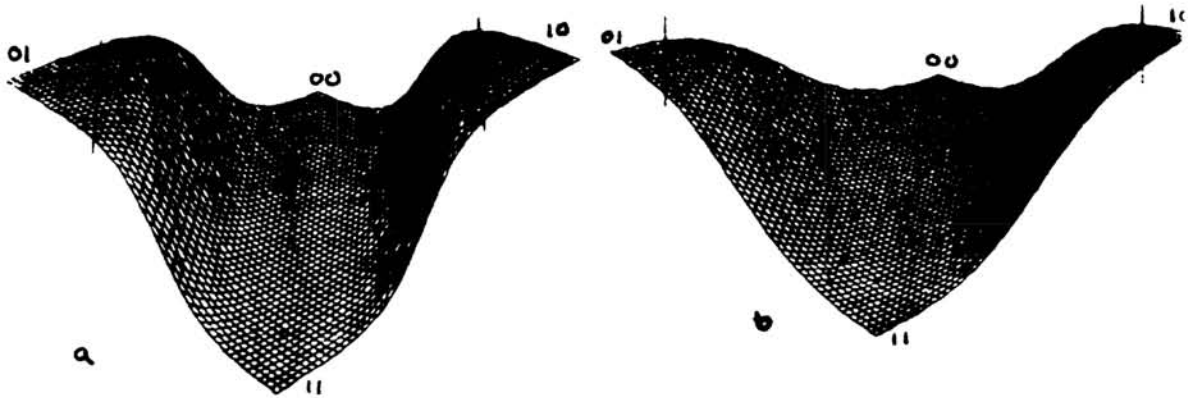

*Figure 4: XOR solved with r=2 (4a) and r=4 (4b)*

solution involves only three cuts in feature space to separate the two "meshed" clusters (see Figure 5a).

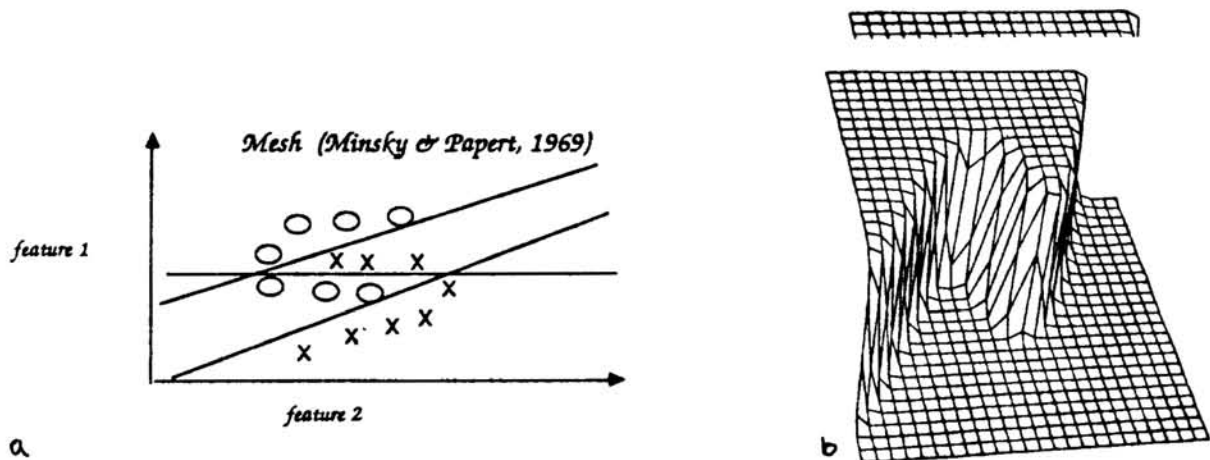

*Figure 5: Mesh problem with minimum cut solution (5a) and Performance Surface(5b)*

Typical solutions for r=2 in this case tend to use a large number of hidden units to separate the two sets of exemplars (see Figure 5b for a performance surface). For example, in Figure 6a notice that a typical (based on several runs) Euclidian back-prop starting with 16 hidden units has found a solution involving five decision boundaries (lines shown in the plane also representing hidden units) while the r=3 case used primarily three decision boundaries and placed a number of other

boundaries redundantly near the center of the meshed region (see Figure 6b) where there is maximum uncertainty about the cluster identification.

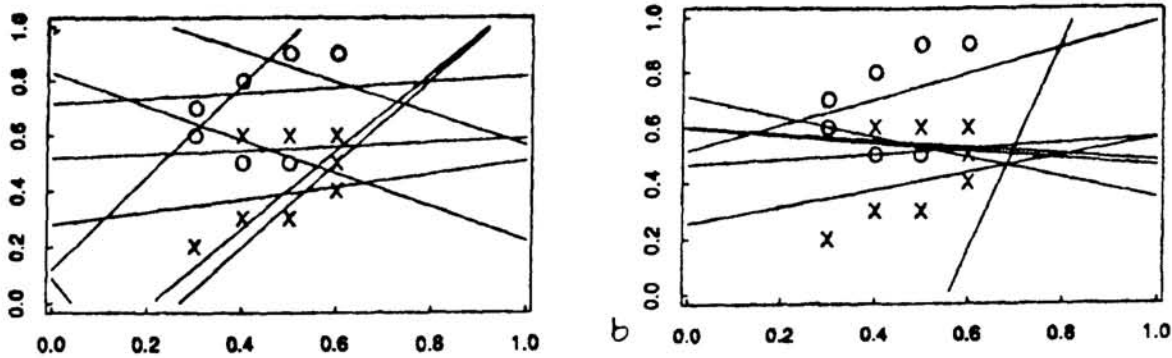

*Figure 6: Mesh solved with r=2 (6a) and r=3 (6b)*

*Speech Recognition.* A final case in which large r's may be appropriate is data that has been previously processed with a transformation that produced compact regions requiring separation in the feature space. One example we have looked at involves spoken digit recognition. The first 10 cepstral coefficients of spoken digits ("one" through "ten") were used for input to a network. In this case an advantage is shown for larger r's with smaller training set sizes. Shown in Figure 7 are transfer data for 50 spoken digits replicated in ten different runs per point (bars show standard error of the mean). Transfer shows a training set size effect for both r=2 and r=3, however for the larger r value at smaller training set sizes (10 and 20) note that transfer is enhanced.

We speculate that this may be due to the larger r backprop creating discrimination regions that are better able to capture the compactness of the clusters inherent in a small number of training points.

## 4. Convergence Properties

It should be generally noted that as r increases, convergence time tends to grow roughly linearly (although this may be problem dependent). Consequently, decreasing r can significantly improve convergence, without much change to the nature of solution. Further, if noise is present decreasing r may reduce it dramatically. Note finally that the gradient for the Minkowski-r back-propagation is nonlinear and therefore more complex for implementing learning procedures.

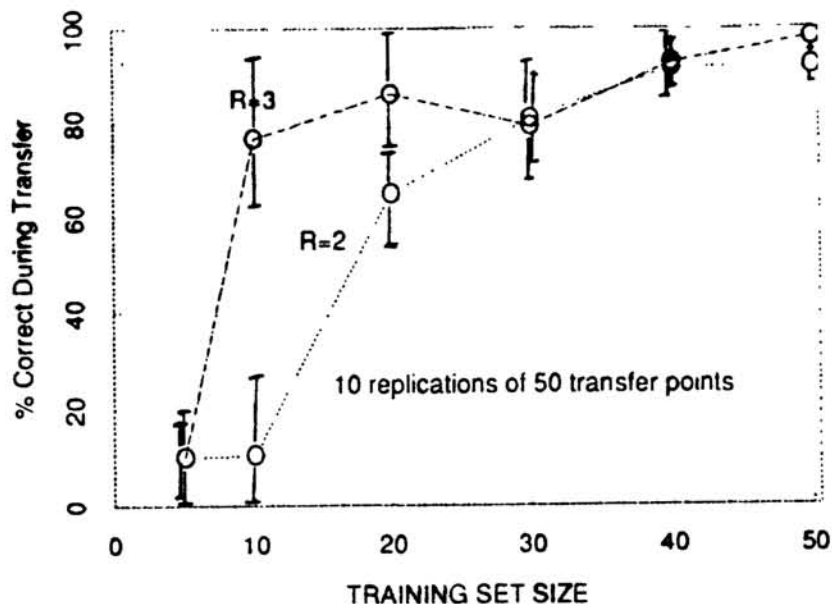

*Figure 7: Digit Recognition Set Size Effect*

## 5. Summary and Conclusion

A new procedure which is a variation on the Back-propagation algorithm is derived and simulated in a number of different problem domains. Noise in the target domain may be reduced by using power values less than 2 and the sensitivity of partition planes to the geometry of the problem may be increased with increasing power values. Other types of objective functions should be explored for their potential consequences on network resources and ensuing pattern recognition capabilities.

## References

1. Rumelhart D. E., Hinton G. E., Williams R., Learning Internal Representations by error propagation. Nature, 1986.

2. Burr D. J. and Hanson S. J., Knowledge Representation in Connectionist Networks, Bellcore, Technical Report,

3. White, H. Personal Communication, 1987.

4. White, H. Some Asymptotic Results for Learning in Single Hidden Layer Feedforward Network Models, Unpublished Manuscript, 1987.

5. Mosteller, F. & Tukey, J. Robust Estimation Procedures, Addison Wesley, 1980.

6. Tukey, J. Personal Communication, 1987.

7. Minsky, M. & Papert, S., Perceptrons: An Introduction to Computational Geometry, MIT Press, 1969.
